# Combining ICA and top-down attention for robust speech recognition

**Un-Min Bae and Soo-Young Lee**

Department of Electrical Engineering and Computer Science
and Brain Science Research Center
Korea Advanced Institute of Science and Technology
373-1 Kusong-dong, Yusong-gu, Taejon, 305-701, Korea
*bum@neuron.kaist.ac.kr, sylee@ee.kaist.ac.kr*

## Abstract

We present an algorithm which compensates for the mismatches between characteristics of real-world problems and assumptions of independent component analysis algorithm. To provide additional information to the ICA network, we incorporate top-down selective attention. An MLP classifier is added to the separated signal channel and the error of the classifier is backpropagated to the ICA network. This backpropagation process results in estimation of expected ICA output signal for the top-down attention. Then, the unmixing matrix is retrained according to a new cost function representing the backpropagated error as well as independence. It modifies the density of recovered signals to the density appropriate for classification. For noisy speech signal recorded in real environments, the algorithm improved the recognition performance and showed robustness against parametric changes.

## 1  Introduction

Independent Component Analysis (ICA) is a method for blind signal separation. ICA linearly transforms data to be statistically as independent from each other as possible [1,2,5]. ICA depends on several assumptions such as linear mixing and source independence which may not be satisfied in many real-world applications. In order to apply ICA to most real-world problems, it is necessary either to release of all assumptions or to compensate for the mismatches with another method.

In this paper, we present a complementary approach to compensate for the mismatches. The top-down selective attention from a classifier to the ICA network provides additional information of the signal-mixing environment. A new cost function is defined to retrain the unmixing matrix of the ICA network considering the propagated information. Under a stationary mixing environment, the averaged adaptation by iterative feedback operations can adjust the feature space to be more helpful to classification performance. This process can be regarded as a selective attention model in which input patterns are adapted according to top-down infor-

mation. The proposed algorithm was applied to noisy speech recognition in real environments and showed the effectiveness of the feedback operations.

## 2 The proposed algorithm

### 2.1 Feedback operations based on selective attention

As previously mentioned, ICA supposes several assumptions. For example, one assumption is a linearly mixing condition, but in general, there is inevitable non-linearity of microphones to record input signals. Such mismatches between the assumptions of ICA and real mixing conditions cause unsuccessful separation of sources. To overcome this problem, a method to supply valuable information to the ICA network was proposed. In the learning phase of ICA, the unmixing matrix is subject to the signal-mixing matrix, not the input patterns. Under stationary mixing environment where the mixing matrix is fixed, iteratively providing additional information of the mixing matrix can contribute to improving blind signal separation performance. The algorithm performs feedback operations from a classifier to the ICA network in the test phase, which adapts the unmixing matrices of ICA according to a newly defined measure considering both independence and classification error. This can result in adaptation of input space of the classifier and so improve recognition performance. This process is inspired from the selective attention model [9,10] which calculates expected input signals according to top-down information.

In the test phase, as shown in Figure 1, ICA separates signal and noise, and Mel-frequency cepstral coefficients (MFCCs) extracted as a feature vector are delivered to a classifier, multi-layer perceptron (MLP). After classification, the error function of the classifier is defined as

$$E_{\mathrm{mlp}} = \frac{1}{2} \sum_i (t_{\mathrm{mlp},i} - y_{\mathrm{mlp},i})^2, \tag{1}$$

where $t_{\mathrm{mlp},i}$ is target value of the output neuron $y_{\mathrm{mlp},i}$. In general, the target values are not known and should be determined from the outputs $y_{\mathrm{mlp}}$. Only the target value of the highest output is set to 1, and the others are set to -1 when the nonlinear function of the classifier is the bipolar sigmoid function. The algorithm performs gradient-descent calculation by error backpropagation. To reduce the error, it computes the required changes of the input values of the classifier and finally those of the unmixed signals of the ICA network. Then, the leaning rule of the ICA algorithm should be changed considering these variations. The newly defined cost function of the ICA network includes the error backpropagated term as well as the joint entropy $H(\mathrm{y_{ica}})$ of the outputs $\mathrm{y_{ica}}$.

$$
\begin{aligned}
E_{\mathrm{ica}} &= -H(\mathrm{y_{ica}}) + \gamma \cdot \frac{1}{2}(\mathrm{u_{target}} - \mathrm{u})(\mathrm{u_{target}} - \mathrm{u})^H \\
&= -H(\mathrm{y_{ica}}) + \gamma \cdot \frac{1}{2}\Delta\mathrm{u}\Delta\mathrm{u}^H, \tag{2}
\end{aligned}
$$

where u are the estimate recovered sources and $\gamma$ is a coefficient which represents the relative importance of two terms. The learning rule derived using gradient descent on the cost function in Eq.(2) is

$$\Delta W \propto [I - \varphi(\mathrm{u})\mathrm{u}^H]W + \gamma \cdot \mathrm{x}\Delta\mathrm{u}, \tag{3}$$

where x are the input signals of the ICA network. The first term in Eq.(3) is the learning rule of ICA which is applicable to complex-valued data in the frequency

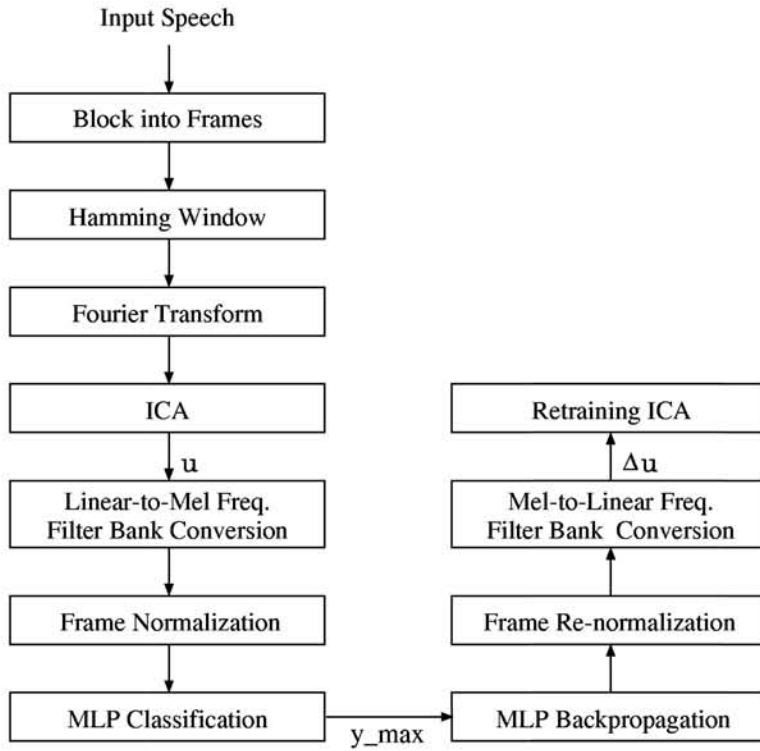

Figure 1: Real-world speech recognition with feedback operations from a classifier to the ICA network

domain [8,11]. In real environments where substantial time delays occur, the observed input signals are convolved mixtures of sources, not linear mixtures and the mixing model no longer is a matrix. In this case, blind signal separation using ICA can be achieved in the frequency domain. The complex score function is

$$\varphi(z) = \tanh(Re\{z\}) + j \cdot \tanh(Im\{z\}). \tag{4}$$

The procedure in the test phase is summarized as follows.

1. For a test input, perform the forward operation and classify the pattern.

2. Define the error function of the classifier in Eq.(1) and perform error backpropagation to find the required changes of unmixed signals of ICA.

3. Define the cost function of the ICA network in Eq.(2) and update unmixing matrix with the learning rule in Eq.(3). Then, go to step 1.

The newly defined error function of the classifier in Eq.(1) does not cause overfitting problems because it is used for updating the unmixing matrix of ICA only once. If classification performance is good, the averaged changes of the unmixing matrix over the total input patterns can contribute to improving recognition performance.

## 2.2 Considering the assumptions of ICA

The assumptions of ICA [3,4,5] are summarized as follows.

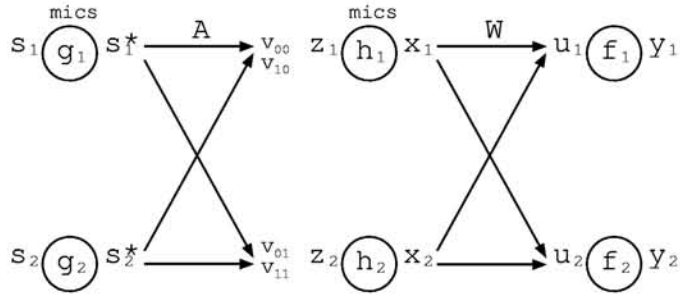

Figure 2: a nonlinear mixing model due to the distortions of microphones

1. The sources are linearly mixed.
2. The sources are mutually independent.
3. At most, one source is normally distributed.
4. The number of sensors is equal to or greater than the number of sources.
5. No sensor noise or only low additive noise signals are permitted.

The assumptions 4 and 5 can be released if there are enough sensors. The assumption 3 is also negligible because the source distribution is usually approximated as super-Gaussian or Laplacian distributions in the speech recognition problem.

As to speech recognition in real mixing environments, the nonlinearity of microphones is an inevitable problem. Figure 2 shows a nonlinear mixing model, the nonlinear functions $g(\cdot)$ and $h(\cdot)$ denote the distortions of microphones. s are original sources, x are observed signals, and u are the estimates of the recovered sources. If the sources $s_1$ and $s_2$ are mutually independent, the random variables $s_1^*$ and $s_2^*$ are still independent each other, and so are $v_{00}$ and $v_{10}$. The density of $z_1 = v_{00} + v_{10}$ equals the convolution of the densities of $v_{00}$ and $v_{10}$ [7].

$$p(z_1) = \int p_{v_{00}}(z_1 - v_{10}) p_{v_{10}}(v_{10}) dv_{10},$$

$$p(x_1) = \frac{p(z_1)}{h_1'}. \tag{5}$$

After all, the observed signal $x_1$ is not a linear mixture of two independent components due to the nonlinear distortion $h(\cdot)$. The assumption of source independence is violated. In this situation, it is hard to expect what would be the ICA solution and to assert the solution is reliable. Even if $x_1$ has two independent components, which is the case of linear distortion of microphones, there is a conflict between independence and source density approximation because the densities of independent components of observed signals are different from those of original sources by $g(\cdot)$ and $h(\cdot)$, and may be far from the density approximated by $f(\cdot)$.

The proposed algorithm can be a solution to this problem. In the training phase, a classifier learns noiseless data and the density of $x_1$ used for the learning is

$$p(x_1) = a_{00} \frac{p(s_1)}{h_1' g_1'}. \tag{6}$$

The second backpropagated term in the cost function Eq.(2) changes the unmixing matrix W to adapt the density of unmixed signals to the density that the classifier

Table 1: The recognition rates of noisy speech recorded with F-16 fighter noise (%)

| SNR | Training data | | | | Test data | | | |
|---|---|---|---|---|---|---|---|---|
| | Clean | 15dB | 10dB | 5dB | Clean | 15dB | 10dB | 5dB |
| MLP | 99.9 | 93.3 | 73.5 | 42.8 | 96.1 | 84.8 | 63.0 | 36.7 |
| ICA | 99.7 | 97.0 | 91.9 | 78.7 | 93.9 | 90.6 | 85.6 | 68.9 |
| **The proposed algorithm** | **99.9** | **99.3** | **94.5** | **80.6** | **96.1** | **93.5** | **86.3** | **71.1** |

learned. This can be a clue to what should be the ICA solution. Iterative operations over the total data induce that the averaged change of the unmixing matrix becomes roughly a function of the nonlinearity $g(\cdot)$ and $h(\cdot)$, not a certain density $p(s_1)$ subject to every pattern.

## 3   Noisy Speech Recognition in Real Environments

The proposed algorithm was applied to isolated-word speech recognition. The input data are convolved mixtures of speech and noise recorded in real environments. The speech data set consists of 75 Korean words of 48 speakers, and F-16 fighter noise and speech babbling noise were used as noise sources. Each ICA network has two inputs and two outputs for the signal and noise sources. Tables 1 and 2 show the recognition results for the three methods: MLP only, MLP with standard ICA, and the proposed algorithm. 'Training data' mean the data used for learning of the classifier, and 'Test data' are the rest. ICA improves classification performance compared to MLP only in the heavy-noise cases, but in the cases of clean data, ICA does not contribute to recognition and the recognition rates are lower than those of MLP only. The proposed algorithm shows better recognition performance than standard ICA for both training and test data. Especially, for the clean data, the proposed algorithm improves the recognition rates to be the same as those of MLP only in most cases. The algorithm reduces the false recognition rates by about 30% to 80% in comparison with standard ICA when signal to noise ratios (SNRs) are 15dB or higher. With such low noise, the classification performance of MLP is relatively reliable, and MLP can provide the ICA network for helpful information. However, with heavy noise, the recognition rates of MLP sharply decrease, and the error backpropagation can hardly provide valuable information to the ICA network. The overall improvement for the training data is higher than that for the test data. This is because the the recognition performance of MLP is better for the training data.

As shown in Figure 3, iterative feedback operations decrease the false recognition rates, and the variation of the unknown parameter $\gamma$ in Eq.(2) doesn't affect the final recognition performance. The variation of the learning rate for updating the unmixing matrix also doesn't affect the final performance, and it only influences on the converging time to reach the final recognition rates. The learning rate was fixed regardless of SNR in all of the experiments.

## 4   Discussion

The proposed algorithm is an approach to complement ICA by providing additional information based on top-down selective attention with a pre-trained MLP classifier. The error backpropagation operations adapt the density of recovered signals

Table 2: The recognition rates of noisy speech recorded with speech babbling noise (%)

| SNR | Training data | | | | Test data | | | |
|---|---|---|---|---|---|---|---|---|
| | Clean | 15dB | 10dB | 5dB | Clean | 15dB | 10dB | 5dB |
| MLP | 99.7 | 88.6 | 61.5 | 32.6 | 96.8 | 82.9 | 64.5 | 38.5 |
| ICA | 98.5 | 95.2 | 91.9 | 76.5 | 91.7 | 88.6 | 85.1 | 73.2 |
| **The proposed algorithm** | **99.7** | **97.7** | **92.5** | **76.7** | **97.2** | **93.1** | **87.4** | **73.4** |

according to the new cost function of ICA. This can help ICA find the solution proper for classification under the nonlinear and independence violations, but this needs the stationary condition. For nonstationary environments, a mixture model like the ICA mixture model [6] can be considered. The ICA mixture model can assign class membership to each environment category and separate independent sources in each class. To completely settle the nonlinearity problem in real environment, it is necessary to introduce a scheme which models the nonlinearity such as the distortions of microphones. Multi-layered ICA can be an approach to model nonlinearity.

In the noisy recognition problem, the proposed algorithm improved recognition performance compared to ICA alone. Especially in moderate noise cases, the algorithm remarkably reduced the false recognition rates. This is due to the high classification performance of the pre-trained MLP. In the case of heavy noise the expected ICA output estimated from the top-down attention may not be accurate, and the selective attention does not help much. It is natural that we only put attention to familiar subjects. Therefore more robust classifiers may be needed for signals with heavy noise.

## Acknowledgments

This work was supported as a Brain Science & Engineering Research Program sponsored by Korean Ministry of Science and Technology.

## References

[1] Amari, S., Cichocki, A., and Yang, H. (1996) A new learning algorithm for blind signal separation, In *Advances in Neural Information Processing Systems 8*, pp. 757-763.

[2] Bell, A. J. and Sejnowski, T. J. (1995) An information-maximization approach to blind separation and blind deconvolution, *Neural Computation*, 7:1129-1159.

[3] Cardoso, J.-F. and Laheld, B. (1996) Equivariant adaptive source separation, *IEEE Trans. on S.P.*, 45(2):434-444.

[4] Comon, P. (1994) Independent component analysis - a new concept?, *Signal Processing*, 36(3):287-314.

[5] Lee, T.-W. (1998) *Independent component analysis - theory and applications*, Kluwer Academic Publishers, Boston.

[6] Lee, T.-W., Lewicki, M. S., and Sejnowski, T. J. (1999) ICA mixture models for unsupervised classification of non-Gaussian sources and automatic context

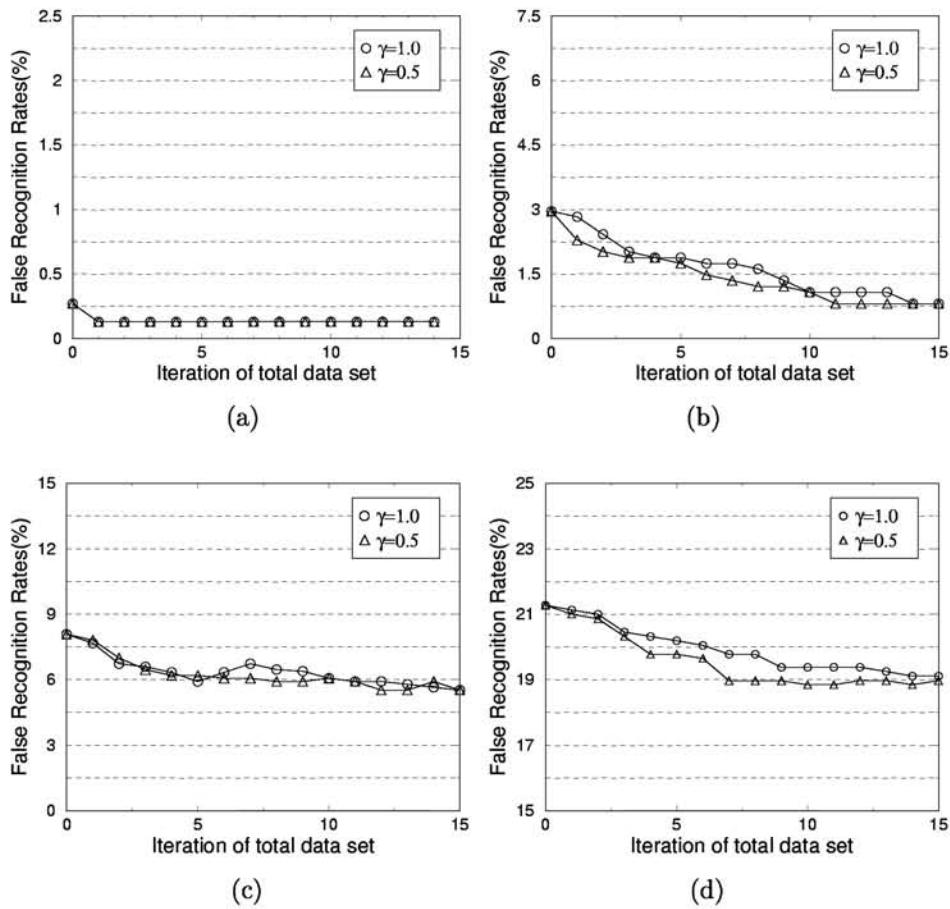

Figure 3: The false recognition rates by iteration of total data and the value of the $\gamma$ parameter. (a) Clean speech; (b) SNR=15 dB; (c) SNR=10 dB; (d) SNR=5 dB

switching in blind signal separation, *IEEE Trans. on Pattern Analysis and Machine Intelligence*, in press.

[7] Papoulis, A. (1991) *Probability, random variables, and stochastic processes*, McGraw-Hill, Inc.

[8] Park, H.-M., Jung, H.-Y., Lee, T.-W., and Lee, S.-Y. (1999) Subband-based blind signal separation for noisy speech recognition, *Electronics Letters*, 35(23):2011-2012.

[9] Park, K.-Y. and Lee, S.-Y. (1999) Selective attention for robust speech recognition in noisy environments, In *Proc. of IJCNN*, paper no. 829.

[10] Park, K.-Y. and Lee, S.-Y. (2000) Out-of-vocabulary rejection based on selective attention model, *Neural Processing Letters*, 12:41-48.

[11] Smaragdis, P. (1997) Information theoretic approaches to source separation, *Masters Thesis*, MIT Media Arts and Science Dept.
